# Minimax embeddings

**Matthew Brand**
Mitsubishi Electric Research Labs
Cambridge MA 02139 USA

## Abstract

Spectral methods for nonlinear dimensionality reduction (NLDR) impose a neighborhood graph on point data and compute eigenfunctions of a quadratic form generated from the graph. We introduce a more general and more robust formulation of NLDR based on the singular value decomposition (SVD). In this framework, most spectral NLDR principles can be recovered by taking a subset of the constraints in a quadratic form built from local nullspaces on the manifold. The minimax formulation also opens up an interesting class of methods in which the graph is "decorated" with information at the vertices, offering discrete or continuous maps, reduced computational complexity, and immunity to some solution instabilities of eigenfunction approaches. Apropos, we show almost all NLDR methods based on eigenvalue decompositions (EVD) have a solution instability that increases faster than problem size. This pathology can be observed (and corrected via the minimax formulation) in problems as small as $N < 100$ points.

## 1 Nonlinear dimensionality reduction (NLDR)

Spectral NLDR methods are graph embedding problems where a set of $N$ points $\mathbf{X} \doteq [\mathbf{x}_1, \cdots, \mathbf{x}_N] \in \mathbb{R}^{D \times N}$ sampled from a low-dimensional manifold in a ambient space $\mathbb{R}^D$ is reparameterized by imposing a neighborhood graph $\mathcal{G}$ on $\mathbf{X}$ and embedding the graph with minimal distortion in a "parameterization" space $\mathbb{R}^d$, $d < D$. Typically the graph is sparse and local, with edges connecting points to their immediate neighbors. The embedding must keep these edges short or preserve their length (for isometry) or angles (for conformality). The graph-embedding problem was first introduced as a least-squares problem by Tutte [1], and as an eigenvalue problem by Fiedler [2]. The use of sparse graphs to generate metrics for least-squares problems has been studied intensely in the following three decades (see [3]). Modern NLDR methods use graph constraints to generate a metric in a space of embeddings $\mathbb{R}^N$. Eigenvalue decomposition (EVD) gives the directions of least or greatest variance under this metric. Typically a subset of $d$ extremal eigenvectors gives the embedding of $N$ points in $\mathbb{R}^d$ parameterization space. This includes the IsoMap family [4], the locally linear embedding (LLE) family [5,6], and Laplacian methods [7,8]. Using similar methods, the Automatic Alignment [6] and Charting [9] algorithms embed local subspaces instead of points, and by combining subspace projections thus obtain continuous maps between $\mathbb{R}^D$ and $\mathbb{R}^d$.

This paper introduces a general algebraic framework for computing optimal embeddings directly from graph constraints. The aforementioned methods can can be recovered as special cases. The framework also suggests some new methods with very attractive properties, including continuous maps, reduced computational complexity, and control over the degree

of conformality/isometry in the desired map. It also eliminates a solution instability that is intrinsic to EVD-based approaches. A perturbational analysis quantifies the instability.

## 2  Minimax theorem for graph embeddings

We begin with neighborhood graph specified by a nondiagonal weighted adjacency matrix $\mathbf{M} \in \mathbb{R}^{N \times N}$ that has the *data-reproducing property* $\mathbf{XM} = \mathbf{X}$ (this can be relaxed to $\mathbf{XM} \approx \mathbf{X}$ in practice). The graph-embedding and NLDR literatures offer various constructions of $\mathbf{M}$, each appropriate to different sets of assumptions about the original embedding and its sampling $\mathbf{X}$ (e.g., isometry, local linearity, noiseless samples, regular sampling, etc.). Typically $M_{ij} \neq 0$ if points $i, j$ are nearby on the intrinsic manifold and $|M_{ij}|$ is small or zero otherwise. Each point is taken to be a linear or convex combination of its neighbors, and thus $\mathbf{M}$ specifies manifold connectivity in the sense that any nondegenerate embedding $\mathbf{Y}$ that satisfies $\mathbf{YM} \approx \mathbf{Y}$ with small residual $\|\mathbf{YM} - \mathbf{Y}\|_F$ will preserve this connectivity and the structure of local neighborhoods. For example, in barycentric embeddings, each point is the average of its neighbors and thus $M_{ij} = 1/k$ if vertex $i$ is connected to vertex $j$ (of degree $k$). We will also consider three optional constraints on the embedding :

1. A null-space restriction, where the solution must be outside to the column-space of $\mathbf{C} \in \mathbb{R}^{N \times M}$, $M < N$. For example, it is common to stipulate that the solution $\mathbf{Y}$ be centered, i.e., $\mathbf{YC} = \mathbf{0}$ for $\mathbf{C} = \mathbf{1}$, the constant vector.

2. A basis restriction, where the solution must be a linear combination of the rows of basis $\mathbf{Z} \in \mathbb{R}^{K \times N}$, $K \leq N$. This can be thought of as information placed at the vertices of the graph that serves as example inputs for a target NLDR function. We will use this to construct dimension-reducing radial basis function networks.

3. A metric $\Sigma \in \mathbb{R}^{N \times N}$ that determines how error is distributed over the points. For example, it might be important that boundary points have less error. We assume that $\Sigma$ is symmetric positive definite and has factorization $\Sigma = \mathbf{AA}^\top$ (e.g., $\mathbf{A}$ could be a Cholesky factor of $\Sigma$).

In most settings, the optional matrices will default to the identity matrix. In this context, we define the per-dimension embedding error of row-vector $\mathbf{y}_i \in \text{rows}(\mathbf{Y})$ to be

$$\mathcal{E}_{\mathbf{M}}(\mathbf{y}_i) \doteq \max_{\mathbf{y}_i \in \text{range}(\mathbf{Z}),, \mathbf{K} \in \mathbb{R}^{M \times N}} \frac{\|(\mathbf{y}_i(\mathbf{M} + \mathbf{CD}) - \mathbf{y}_i)\mathbf{A}\|}{\|\mathbf{y}_i \mathbf{A}\|} \tag{1}$$

where $\mathbf{D}$ is a matrix constructed by an adversary to maximize the error. The optimizing $\mathbf{y}_i$ is a vector inside the subspace spanned by the rows of $\mathbf{Z}$ and outside the subspace spanned by the columns of $\mathbf{C}$, for which the reconstruction residual $\mathbf{y}_i \mathbf{M} - \mathbf{y}_i$ has smallest norm w.r.t. the metric $\Sigma$. The following theorem identifies the optimal embedding $\mathbf{Y}$ for any choice of $\mathbf{M}, \mathbf{Z}, \mathbf{C}, \Sigma$:

**Minimax solution:** Let $\mathbf{Q} \in \mathbb{S}^{K \times P}$ be a column-orthonormal basis of the null-space of the rows of $\mathbf{ZC}$, with $P = K - \text{rank}(\mathbf{C})$. Let $\mathbf{B} \in \mathbb{R}^{P \times P}$ be a square factor satisfying $\mathbf{B}^\top \mathbf{B} = \mathbf{Q}^\top \mathbf{Z} \Sigma \mathbf{Z}^\top \mathbf{Q}$, e.g., a Cholesky factor (or the "R" factor in QR-decomposition of $(\mathbf{Q}^\top \mathbf{ZA})^\top$). Compute the left singular vectors $\mathbf{U} \in \mathbb{S}^{N \times N}$ of $\mathbf{U} \text{diag}(\mathbf{s})\mathbf{V}^\top = \mathbf{B}^{-\top} \mathbf{Q}^\top \mathbf{Z}(\mathbf{I} - \mathbf{M})\mathbf{A}$, with singular values $\mathbf{s}^\top \doteq [s_1, \cdots, s_P]$ ordered $s_1 \leq s_2 \leq \cdots \leq s_p$. Using the leading columns $\mathbf{U}_{1:d}$ of $\mathbf{U}$, set $\mathbf{Y} = \mathbf{U}_{1:d}^\top \mathbf{B}^{-\top} \mathbf{Q}^\top \mathbf{Z}$.

**Theorem 1.** $\mathbf{Y}$ *is the optimal (minimax) embedding in* $\mathbb{R}^d$ *with error* $\|[s_1, \cdots, s_d]\|^2$:

$$\mathbf{Y} \doteq \mathbf{U}_{1:d}^\top \mathbf{B}^{-\top} \mathbf{Q}^\top \mathbf{Z} = \arg \min_{\mathbf{Y} \in \mathbb{R}^{d \times N}} \sum_{\mathbf{y}_i \in rows(\mathbf{Y})} \mathcal{E}_{\mathbf{M}}(\mathbf{y}_i)^2 \quad with \quad \mathcal{E}_{\mathbf{M}}(\mathbf{y}_i) = s_i. \tag{2}$$

Appendix A develops the proof and other error measures that are minimized.

Local NLDR techniques are easily expressed in this framework. When $\mathbf{Z} = \mathbf{A} = \mathbf{I}$, $\mathbf{C} = []$, and $\mathbf{M}$ reproduces $\mathbf{X}$ through linear combinations with $\mathbf{M}^\top \mathbf{1} = \mathbf{1}$, we recover LLE [5]. When $\mathbf{Z} = \mathbf{I}$, $\mathbf{C} = []$, $\mathbf{I} - \mathbf{M}$ is the normalized graph Laplacian, and $\mathbf{A}$ is a diagonal matrix of vertex degrees, we recover Laplacian eigenmaps [7]. When further $\mathbf{Z} = \mathbf{X}$ we recover locally preserving projections [8].

## 3   Analysis and generalization of charting

The minimax construction of charting [9] takes some development, but offers an interesting insight into the above-mentioned methods. Recall that charting first solves for a set of local affine subspace axes $\mathbf{S}_1 \in \mathbb{R}^{D \times d}, \mathbf{S}_2, \cdots$ at offsets $\mu_1 \in \mathbb{R}^D, \mu_2, \cdots$ that best cover the data and vary smoothly over the manifold. Each subspace offers a chart—a local parameterization of the data by projection onto the local axes. Charting then constructs a weighted mixture of affine projections that merges the charts into a global parameterization. If the data manifold is curved, each projection will assign a point a slightly different embedding, so the error is measured as the variance of these proposed embeddings about their mean. This maximizes consistency and tends to produce isometric embeddings; [9] discusses ways to explicitly optimize the isometry of the embedding.

Under the assumption of isometry, the charting error is equivalent to the sum-squared displacements of an embedded point relative to its immediate neighbors (summed over all neighborhoods). To construct the same error criteria in the minimax setting, let $\mathbf{x}_{i-k}, \cdots, \mathbf{x}_i, \cdots, \mathbf{x}_{i+k}$ denote points in the $i^{\text{th}}$ neighborhood and let the columns of $\mathbf{V}_i \in \mathbb{R}^{(2k+1) \times d}$ be an orthonormal basis of rows of the local parameterization $\mathbf{S}_i^\top [\mathbf{x}_{i-k}, \cdots, \mathbf{x}_i, \cdots, \mathbf{x}_{i+k}]$. Then a nonzero reparameterization will satisfy $[\mathbf{y}_{i-k}, \cdots, \mathbf{y}_i, \cdots, \mathbf{y}_{i+k}] \mathbf{V}_i \mathbf{V}_i^\top = [\mathbf{y}_{i-k}, \cdots, \mathbf{y}_i, \cdots, \mathbf{y}_{i+k}]$ if and only if it preserves the relative position of the points in the local parameterization. Conversely, any relative displacements of the points are isolated by the formula $[\mathbf{y}_{i-k}, \cdots, \mathbf{y}_i, \cdots, \mathbf{y}_{i+k}](\mathbf{I} - \mathbf{V}_i \mathbf{V}_i^\top)$. Minimizing the Frobenius norm of this expression is thus equivalent to minimizing the local error in charting. We sum these constraints over all neighborhoods to obtain the constraint matrix $\mathbf{M} = \mathbf{I} - \sum_i \mathbf{F}_i (\mathbf{I} - \mathbf{V}_i \mathbf{V}_i^\top) \mathbf{F}_i^\top$, where $(\mathbf{F}_i)_{kj} = 1$ iff the $j^{\text{th}}$ point of the $i^{\text{th}}$ neighborhood is the $k^{\text{th}}$ point of the dataset. *Because $\mathbf{V}_i \mathbf{V}_i^\top$ and $(\mathbf{I} - \mathbf{V}_i \mathbf{V}_i^\top)$ are complementary, it follows that the error criterion of any local NLDR method (e.g., LLE, Laplacian eigenmaps, etc.) must measure the projection of the embedding onto some* subspace *of* $(\mathbf{I} - \mathbf{V}_i \mathbf{V}_i^\top)$.

To construct a continuous map, charting uses an overcomplete radial basis function (RBF) representation $\mathbf{Z} = [z(\mathbf{x}_1), z(\mathbf{x}_2), \cdots z(\mathbf{x}_N)]$, where $z(\mathbf{x})$ is a vector that stacks $z_1(\mathbf{x})$, $z_2(\mathbf{x})$, etc., and

$$z_m(\mathbf{x}) \doteq \begin{bmatrix} \mathbf{K}_m^\top (\mathbf{x} - \mu_m) \\ 1 \end{bmatrix} \frac{p_m(\mathbf{x})}{\sum_m p_m(\mathbf{x})}, \tag{3}$$

$$p_m(\mathbf{x}) \doteq \mathcal{N}(\mathbf{x} | \mu_m, \Sigma_m) \propto e^{-(\mathbf{x} - \mu_m)^\top \Sigma_m^{-1} (\mathbf{x} - \mu_m)/2} \tag{4}$$

and $\mathbf{K}_m$ is any local linear dimensionality reducer, typically $\mathbf{S}_m$ itself. Each column of $\mathbf{Z}$ contains many "views" of the same point that are combined to give its low-dimensional embedding.

Finally, we set $\mathbf{C} = \mathbf{1}$, which forces the embedding of the full data to be centered.

Applying the minimax solution to these constraints yields the RBF network mixing matrix, $f(\mathbf{x}) \doteq \mathbf{U}_{1:d}^\top \mathbf{B}^{-\top} \mathbf{Q}^\top z(\mathbf{x})$. Theorem 1 guarantees that the resulting embedding is least-squares optimal w.r.t. $\mathbf{Z}, \mathbf{M}, \mathbf{C}, \mathbf{A}$ at the datapoints $f(\mathbf{x}_i)$, and because $f(\cdot)$ is an affine transform of $z(\cdot)$ it smoothly interpolates the embedding between points.

There are some interesting variants:

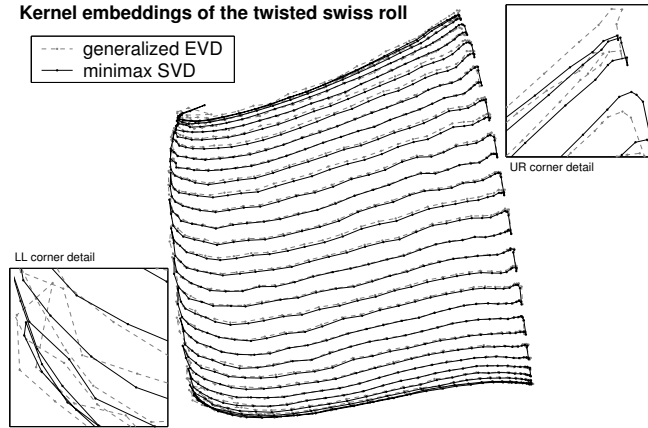

**Kernel embeddings of the twisted swiss roll**

- - - generalized EVD
——— minimax SVD

UR corner detail

LL corner detail

**Fig. 1.** Minimax and generalized EVD solution for kernel eigenmap of a non-developable swiss roll. Points are connected into a grid which ideally should be regular. The EVD solution shows substantial degradation. Insets detail corners where the EVD solution crosses itself repeatedly. The border compression is characteristic of Laplacian constraints.

**One-shot charting:** If we set the local dimensionality reducers to the identity matrix (all $\mathbf{K}_m = \mathbf{I}$), then the minimax method jointly optimizes the local dimensionality reduction to charts and the global coordination of the charts (under any choice of $\mathbf{M}$). This requires that rows($\mathbf{Z}$) $\leq N$ for a fully determined solution.

**Discrete isometric charting:** If $\mathbf{Z} = \mathbf{I}$ then we directly obtain a discrete isometric embedding of the data, rather than a continuous map, making this a local equivalent of IsoMap.

**Reduced basis charting:** Let $\mathbf{Z}$ be constructed using just a small number of kernels randomly placed on the data manifold, such that rows($\mathbf{Z}$) $\ll N$. Then the size of the SVD problem is substantially reduced.

## 4  Numerical advantage of minimax method

Note that the minimax method projects the constraint matrix $\mathbf{M}$ into a subspace derived from $\mathbf{C}$ and $\mathbf{Z}$ and decomposes it there. This suppresses unwanted degrees of freedom (DOFs) admitted by the problem constraints, for example the trivial $\mathbb{R}^0$ embedding where all points are mapped to a single point $y_i = N^{-1/2}$. The $\mathbb{R}^0$ embedding serves as a translational DOF in the solution. LLE- and eigenmap-based methods construct $\mathbf{M}$ to have a constant null-space so that the translational DOF will be isolated in the EVD as null eigenvalue paired to a constant eigenvector, which is then discarded. However, section 4.1 shows that this construction makes the EVD increasingly unstable as problem size grows and/or the data becomes increasing amenable to low-residual embeddings, ultimately causing solution collapse. As the next paragraph demonstrates, the problem is exacerbated when embedding w.r.t. a basis $\mathbf{Z}$ (via the equivalent generalized eigenproblem), partly because the eigenvector associated with the unwanted DOF can have arbitrary structure. In all cases the problem can be averted by using the minimax formulation with $\mathbf{C} = \mathbf{1}$ to suppress the DOF.

A 2D plane was embedded in 3D with a curl, a twist, and 2.5% Gaussian noise, then regularly sampled at 900 points. We computed a kernelized Laplacian eigenmap using 70 random points as RBF centers, i.e., a continous map using $\mathbf{M}$ derived from the graph Laplacian and $\mathbf{Z}$ constructed as above. The map was computed both via the minimax (SVD) method and via the equivalent generalized eigenproblem, where the translational degree of freedom must be removed by discarding an eigenvector from the solution. The two solutions are algebraically equivalent in every other regard. A variety of eigensolvers were tried; we took

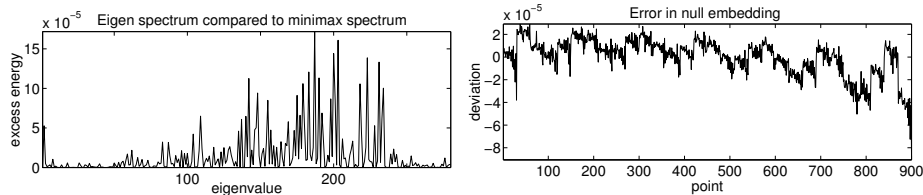

**Fig. 2.** Excess energy in the eigenspectrum indicates that the translational DOF has contaminated many eigenvectors. If the EVD had successfully isolated the unwanted DOF, then its remaining eigenvalues should be identical to those derived from the minimax solution. The graph at left shows the difference in the eigenspectra. The graph at right shows the EVD solution's deviation from the translational vector $\mathbf{y_0} = \mathbf{1} \cdot N^{-1/2} \approx .03333$. If the numerics were perfect the line would be flat, but in practice the deviation is significant enough (roughly 1% of the diameter of the embedding) to noticably perturb points in figure 1.

the best result. Figure 1 shows that the EVD solution exhibits many defects, particularly a folding-over of the manifold at the top and bottom edges and at the corners. Figure 2 shows that the noisiness of the EVD solution is due largely to mutual contamination of numerically unstable eigenvectors.

### 4.1 Numerical instability of eigen-methods

The following theorem uses tools of matrix perturbation theory to show that as the problem size increases, the desired and unwanted eigenvectors become increasingly wobbly and gradually contaminate each other, leading to degraded solutions. More precisely, the low-order eigenvalues are ill-conditioned and exhibit multiplicities that may be true (due to noiseless samples from low-curvature manifolds) or false (due to numerical noise). Although in many cases some post-hoc algebra can "filter" the unwanted components out of the contaminated eigensolution, it is not hard to construct cases where the eigenvectors cannot be cleanly separated. The minimax formulation is immune to this problem because it explicitly suppresses the gratuitous component(s) *before* matrix decomposition.

**Theorem 2.** *For any finite numerical precision, as the number of points N increases, the Frobenius norm of numerical noise in the null eigenvector $\mathbf{v}_0$ can grow as $O(N^{3/2})$, and the eigenvalue problem can approach a false multiplicity at a rate as fast as $O(N^{3/2})$, at which point the eigenvectors of interest—embedding and translational—are mutually contaminated and/or have an indeterminate eigenvalue ordering.*

Please see appendix B for the proof. This theorem essentially lower-bounds an upper-bound on error; examples can be constructed in which the problem is worse. For example, it can be shown analytically that when embedding points drawn from the simple curve $\mathbf{x}_i = [a, \cos \pi a]^\top$, $a \in [0, 1]$ with $K = 2$ neighbors, instabilities cannot be bounded better than $O(N^{5/2})$; empirically we see eigenvector mixing with $N < 100$ points and we see it grow at the rate $\approx O(N^4)$—in many different eigensolvers. At very large scales, more pernicious instabilities set in. E.g., by $N = 20000$ points, the solution begins to fold over. Although algebraic multiplicity and instability of the eigenproblem is conceptually a minor oversight in the algorithmic realizations of eigenfunction embeddings, as theorem 2 shows, the consequences are eventually fatal.

## 5   Summary

One of the most appealing aspects of the spectral NLDR literature is that algorithms are usually motivated from analyses of linear operators on smooth differentiable manifolds, e.g., [7]. Understandably, these analysis rely on assumptions (e.g., smoothness or isometry

or noiseless sampling) that make it difficult to predict what algorithmic realizations will do when real, noisy data violates these assumptions. The minimax embedding theorem provides a complete algebraic characterization of this *discrete* NLDR problem, and provides a solution that recovers numerically robustified versions of almost all known algorithms. It offers a principled way of constructing new algorithms with clear optimality properties and good numerical conditioning—notably the construction of a continuous NLDR map (an RBF network) in a one-shot optimization (SVD). We have also shown how to cast several local NLDR principles in this framework, and upgrade these methods to give continuous maps. Working in the opposite direction, we sketched the minimax formulation of isometric charting and showed that its constraint matrix contains a superset of all the algebraic constraints used in local NLDR techniques.

## References

1. W.T. Tutte. How to draw a graph. *Proc. London Mathematical Society*, 13:743–768, 1963.

2. Miroslav Fiedler. A property of eigenvectors of nonnegative symmetric matrices and its application to graph theory. *Czech. Math. Journal*, 25:619–633, 1975.

3. Fan R.K. Chung. *Spectral graph theory*, volume 92 of *CBMS Regional Conference Series in Mathematics*. American Mathematical Society, 1997.

4. Joshua B. Tenenbaum, Vin de Silva, and John C. Langford. A global geometric framework for nonlinear dimensionality reduction. *Science*, 290:2319–2323, December 22 2000.

5. Sam T. Roweis and Lawrence K. Saul. Nonlinear dimensionality reduction by locally linear embedding. *Science*, 290:2323–2326, December 22 2000.

6. Yee Whye Teh and Sam T. Roweis. Automatic alignment of hidden representations. In *Proc. NIPS-15*, 2003.

7. Mikhail Belkin and Partha Niyogi. Laplacian eigenmaps for dimensionality reduction and data representation. volume 14 of *Advances in Neural Information Processing Systems*, 2002.

8. Xiafei He and Partha Niyogi. Locality preserving projections. Technical Report TR-2002-09, University of Chicago Computer Science, October 2002.

9. Matthew Brand. Charting a manifold. volume 15 of *Advances in Neural Information Processing Systems*, 2003.

10. G.W. Stewart and Ji-Guang Sun. *Matrix perturbation theory*. Academic Press, 1990.

## A   Proof of minimax embedding theorem (1)

The burden of this proof is carried by supporting lemmas, below. To emphasize the proof strategy, we give the proof first; supporting lemmas follow.

*Proof.* Setting $\mathbf{y}_i = \mathbf{l}_i^\top \mathbf{Z}$, we will solve for $\mathbf{l}_i \in \text{columns}(\mathbf{L})$. Writing the error in terms of $\mathbf{l}_i$,

$$\mathcal{E}_{\mathbf{M}}(\mathbf{l}_i) = \max_{\mathbf{K} \in \mathbb{R}^{M \times N}} \frac{\|\mathbf{l}_i^\top \mathbf{Z}(\mathbf{I} - \mathbf{M} - \mathbf{CK})\mathbf{A}\|}{\|\mathbf{l}_i^\top \mathbf{ZA}\|} = \max_{\mathbf{K} \in \mathbb{R}^{M \times N}} \frac{\|\mathbf{l}_i^\top \mathbf{Z}(\mathbf{I} - \mathbf{M})\mathbf{A} - \mathbf{l}_i^\top \mathbf{ZCKA}\|}{\|\mathbf{l}_i^\top \mathbf{ZA}\|}. \quad (5)$$

The term $\mathbf{l}_i^\top \mathbf{ZCKA}$ produces infinite error unless $\mathbf{l}_i^\top \mathbf{ZC} = \mathbf{0}$, so we accept this as a constraint and seek

$$\min_{\mathbf{l}_i^\top \mathbf{ZC} = \mathbf{0}} \frac{\|\mathbf{l}_i^\top \mathbf{Z}(\mathbf{I} - \mathbf{M})\mathbf{A}\|}{\|\mathbf{l}_i^\top \mathbf{ZA}\|}. \quad (6)$$

By lemma 1, that orthogonality is satisfied by solving the problem in the space orthogonal to $\mathbf{ZC}$; the basis for this space is given by columns of $\mathbf{Q} \doteq \text{null}((\mathbf{ZC})^\top)$.

By lemma 2, the denominator of the error specifies the metric in solution space to be $\mathbf{ZAA}^\top \mathbf{Z}^\top$; when the problem is projected into the space orthogonal to $\mathbf{ZC}$ it becomes $\mathbf{Q}^\top (\mathbf{ZAA}^\top \mathbf{Z}^\top)\mathbf{Q}$. Nesting the "orthogonally-constrained-SVD" construction of lemma 1

inside the "SVD-under-a-metric" lemma 2, we obtain a solution that uses the correct metric in the orthogonal space:

$$\mathbf{B}^\top \mathbf{B} = \mathbf{Q}^\top \mathbf{Z}\mathbf{A}\mathbf{A}^\top \mathbf{Z}^\top \mathbf{Q} \tag{7}$$

$$\mathbf{U}\text{diag}(\mathbf{s})\mathbf{V}^\top = \mathbf{B}^{-\top}\{\mathbf{Q}(\mathbf{Z}(\mathbf{I}-\mathbf{M})\mathbf{A})\} \tag{8}$$

$$\mathbf{L} = \mathbf{Q}\mathbf{B}^{-1}\mathbf{U} \tag{9}$$

where braces indicate the nesting of lemmas. By the "best-projection" lemma (#3), if we order the singular values by ascending magnitude,

$$\mathbf{L}_{1:d} = \arg \min_{\mathbf{J}\in\mathbb{R}^{N\times d}} \sqrt{\Sigma_{\mathbf{j}_i \in \text{cols}(\mathbf{J})}(\|\mathbf{j}^\top \mathbf{Z}(\mathbf{I}-\mathbf{M})\mathbf{A}\|/\|\mathbf{j}\|_{\mathbf{Z}\Sigma\mathbf{Z}^\top})^2} \tag{10}$$

The proof is completed by making the substitutions $\mathbf{L}^\top \mathbf{Z} \to \mathbf{Y}^\top$ and $\|\mathbf{x}^\top \mathbf{A}\| \to \|\mathbf{x}\|_\Sigma$ (for $\Sigma = \mathbf{A}\mathbf{A}^\top$), and leaving off the final square root operation to obtain

$$(\mathbf{Y}^\top)_{1:d} = \arg \min_{\mathbf{J}\in\mathbb{R}^{N\times d}} \Sigma_{\mathbf{j}_i \in \text{cols}(\mathbf{J})}\left(\|\mathbf{j}^\top(\mathbf{I}-\mathbf{M})\|_\Sigma/\|\mathbf{j}\|_\Sigma\right)^2 . \tag{11}$$

**Lemma 1.** *Orthogonally constrained* SVD*: The left singular vectors $\mathbf{L}$ of matrix $\mathbf{M}$ under the constraint $\mathbf{U}^\top \mathbf{C} = \mathbf{0}$ are calculated as $\mathbf{Q} \doteq null(\mathbf{C}^\top)$, $\mathbf{U}\text{diag}(\mathbf{s})\mathbf{V}^\top \overset{SVD}{\Leftarrow} \mathbf{Q}^\top \mathbf{M}$, $\mathbf{L} = \mathbf{Q}\mathbf{U}$.*

*Proof.* First observe that $\mathbf{L}$ is orthogonal to $\mathbf{C}$: By definition, the null-space basis satisfies $\mathbf{Q}^\top \mathbf{C} = \mathbf{0}$, thus $\mathbf{L}^\top \mathbf{C} = \mathbf{U}^\top \mathbf{Q}^\top \mathbf{C} = \mathbf{0}$. Let $\mathbf{J}$ be an orthonormal basis for $\mathbf{C}$, with $\mathbf{J}^\top \mathbf{J} = \mathbf{I}$ and $\mathbf{Q}^\top \mathbf{J} = \mathbf{0}$. Then $\mathbf{L}\text{diag}(\mathbf{s})\mathbf{V}^\top = \mathbf{Q}\mathbf{Q}^\top \mathbf{M} = (\mathbf{I}-\mathbf{J}\mathbf{J}^\top)\mathbf{M}$, the orthogonal projector of $\mathbf{C}$ applied to $\mathbf{M}$, proving that the SVD captures the component of $\mathbf{M}$ that is orthogonal to $\mathbf{C}$.

**Lemma 2.** SVD *with respect to a metric: The vectors $\mathbf{l}_i \in \mathbf{L}$, $\mathbf{v}_i \in \mathbf{V}$ that diagonalize matrix $\mathbf{M}$ with respect to positive definite column-space metric $\Sigma$ are calculated as $\mathbf{B}^\top \mathbf{B} \leftarrow \Sigma$, $\mathbf{U}\text{diag}(\mathbf{s})\mathbf{V}^\top \overset{SVD}{\Leftarrow} \mathbf{B}^{-\top}\mathbf{M}$, $\mathbf{L} \doteq \mathbf{B}^{-1}\mathbf{U}$ satisfy $\|\mathbf{l}_i^\top \mathbf{M}\|/\|\mathbf{l}_i\|_\Sigma = s_i$ and extremize this form for the extremal singular values $s_{\min}, s_{\max}$.*

*Proof.* By construction, $\mathbf{L}$ and $\mathbf{V}$ diagonalize $\mathbf{M}$:

$$\mathbf{L}^\top \mathbf{M}\mathbf{V} = (\mathbf{B}^{-1}\mathbf{U})^\top \mathbf{M}\mathbf{V} = \mathbf{U}^\top(\mathbf{B}^{-\top}\mathbf{M})\mathbf{V} = \text{diag}(\mathbf{s}) \tag{12}$$

and $\text{diag}(\mathbf{s})\mathbf{V}^\top = \mathbf{B}^{-\top}\mathbf{M}$. Forming the gram matrices of both sides of the last line, we obtain the identity $\mathbf{V}\text{diag}(\mathbf{s})^2\mathbf{V}^\top = \mathbf{M}^\top \mathbf{B}^{-1}\mathbf{B}^{-\top}\mathbf{M} = \mathbf{M}^\top \Sigma^{-1}\mathbf{M}$, which demonstrates that $s_i \in \mathbf{s}$ are the singular values of $\mathbf{M}$ w.r.t. column-space metric $\Sigma$. Finally, $\mathbf{L}$ is orthonormal w.r.t. the metric $\Sigma$, because $\|\mathbf{L}\|_\Sigma^2 = \mathbf{L}^\top \Sigma\mathbf{L} = \mathbf{U}^\top \mathbf{B}^{-\top}\mathbf{B}^\top \mathbf{B}\mathbf{B}^{-1}\mathbf{U} = \mathbf{I}$. Consequently,

$$\|\mathbf{l}^\top \mathbf{M}\|/\|\mathbf{l}\|_\Sigma = \|\mathbf{l}^\top \mathbf{M}\|/1 = \|s_i \mathbf{v}_i^\top\| = s_i . \tag{13}$$

and by the Courant-Hilbert theorem,

$$s_{\max} = \max_{\mathbf{l}} \|\mathbf{l}^\top \mathbf{M}\|/\|\mathbf{l}\|_\Sigma; \qquad s_{\min} = \min_{\mathbf{l}} \|\mathbf{l}^\top \mathbf{M}\|/\|\mathbf{l}\|_\Sigma. \tag{14}$$

**Lemma 3.** *Best projection: Taking $\mathbf{L}$ and $\mathbf{s}$ from lemma 2, let the columns of $\mathbf{L}$ and elements of $\mathbf{s}$ be sorted so that $s_1 \geq s_2 \geq \cdots \geq s_N$. Then for any dimensionality $1 \leq d \leq N$,*

$$\mathbf{L}_{1:d} \doteq [\mathbf{l}_1, \cdots, \mathbf{l}_d] = \arg \max_{\mathbf{J}\in\mathbb{R}^{N\times d}} \|\mathbf{J}^\top \mathbf{M}\|_{(\mathbf{J}^\top \Sigma\mathbf{J})^{-1}} \tag{15}$$

$$= \arg \max_{\mathbf{J}\in\mathbb{R}^{N\times d}|\mathbf{J}^\top \Sigma\mathbf{J}=\mathbf{I}} \|\mathbf{J}^\top \mathbf{M}\|_F \tag{16}$$

$$= \arg \max_{\mathbf{J}\in\mathbb{R}^{N\times d}} \sqrt{\Sigma_{\mathbf{j}_i \in cols(\mathbf{J})}(\|\mathbf{j}^\top \mathbf{M}\|/\|\mathbf{j}\|_\Sigma)^2} \tag{17}$$

*with the optimum value of all right hand sides being $(\Sigma_{i=1}^d s_i^2)^{1/2}$. If the sort order is reversed, the minimum of this form is obtained.*

*Proof.* By the Eckart-Young-Mirsky theorem, if $\mathbf{U}^\top \mathbf{M} \mathbf{V} = \text{diag}(\mathbf{s})$ with singular values sorted in descending order, then $\mathbf{U}_{1:d} \doteq [\mathbf{u}_1, \cdots, \mathbf{u}_d] = \arg\max_{\mathbf{U} \in \mathbb{S}^{N \times d}} \|\mathbf{U}^\top \mathbf{M}\|_F$. We first extend this to a non-orthonogonal basis $\mathbf{J}$ under a Mahalonobis norm:

$$\max_{\mathbf{J} \in \mathbb{R}^{N \times d}} \|\mathbf{J}^\top \mathbf{M}\|_{(\mathbf{J}^\top \mathbf{J})^{-1}} = \max_{\mathbf{U} \in \mathbb{S}^{N \times d}} \|\mathbf{U}^\top \mathbf{M}\|_F \tag{18}$$

because $\|\mathbf{J}^\top \mathbf{M}\|_{(\mathbf{J}^\top \mathbf{J})^{-1}}^2 = \text{trace}(\mathbf{M}^\top \mathbf{J} (\mathbf{J}^\top \mathbf{J})^{-1} \mathbf{J}^\top \mathbf{M}) = \text{trace}(\mathbf{M}^\top \mathbf{J} \mathbf{J}^+ (\mathbf{J} \mathbf{J}^+)^\top \mathbf{M}) = \|(\mathbf{J} \mathbf{J}^+) \mathbf{M}\|_F^2 = \|\mathbf{U} \mathbf{U}^\top \mathbf{M}\|_F^2 = \|\mathbf{U}^\top \mathbf{M}\|_F^2$ since $\mathbf{J} \mathbf{J}^+$ is a (symmetric) orthogonal projector having binary eigenvalues $\lambda \in \{0, 1\}$ and therefore it is the gram of an thin orthogonal matrix. We then impose a metric $\Sigma$ on the column-space of $\mathbf{J}$ to obtain the first criterion (equation 15), which asks what maximizes variance in $\mathbf{J}^\top \mathbf{M}$ while minimizing the norm of $\mathbf{J}$ w.r.t. metric $\Sigma$. Here it suffices to substitute in the leading (resp., trailing) columns of $\mathbf{L}$ and verify that the norm is maximized (resp., minimized). Expanding, $\|\mathbf{L}_{1:d}^\top \mathbf{M}\|_{(\mathbf{L}_{1:d}^\top \Sigma \mathbf{L}_{1:d})^{-1}}^2 = \text{trace}((\mathbf{L}_{1:d}^\top \mathbf{M})^\top (\mathbf{L}_{1:d}^\top \Sigma \mathbf{L}_{1:d})^{-1} (\mathbf{L}_{1:d}^\top \mathbf{M})) = \text{trace}((\mathbf{L}_{1:d}^\top \mathbf{M})^\top \mathbf{I} (\mathbf{L}_{1:d}^\top \mathbf{M})) = \text{trace}((\text{diag}(\mathbf{s}_{1:d}) \mathbf{V}_{1:d}^\top)^\top (\text{diag}(\mathbf{s}_{1:d}) \mathbf{V}_{1:d}^\top)) = \|\mathbf{s}_{1:d}\|^2$. Again, by the Eckart-Young-Mirsky theorem, these are the maximal variance-preserving projections, so the first criterion is indeed maximized by setting $\mathbf{J}$ to the columns in $\mathbf{L}$ corresponding to the largest values in $\mathbf{s}$.

Criterion #2 restates the first criterion with the set of candidates for $\mathbf{J}$ restricted to (the hyperelliptical manifold of) matrices that reduce the metric on the norm to the identity matrix (thereby recovering the Frobenius norm). Criterion #3 criterion merely expands the above trace by individual singular values. Note that the numerator and denominator can have different metrics because they are norms in different spaces, possibly of different dimension. Finally, that the trailing $d$ eigenvectors *minimize* these criteria follows directly from the fact that leading $N - d$ singular values account for the maximal part of the variance.

## B  Proof of instability theorem (2)

*Proof.* When generated from a sparse graph with average degree $K$, weighted connectivity matrix $\mathbf{W}$ is sparse and has $O(NK)$ entries. Since the graph vertices represent samples from a smooth manifold, increasing the sampling density $N$ does not change the distribution of magnitudes in $\mathbf{W}$. Consider a perturbation of the nonzero values in $\mathbf{W}$, e.g., $\mathbf{W} \to \mathbf{W} + \mathbf{E}$ due to numerical noise $\mathbf{E}$ created by finite machine precision. By the weak law of large numbers, the Frobenius norm of the sparse perturbation grows as $\|\mathbf{E}\|_F \sim O(\sqrt{N})$. However the $t^{\text{th}}$-smallest nonzero eigenvalue $\lambda_t(\mathbf{W})$ grows as $\lambda_t(\mathbf{W}) = \mathbf{v}_t^\top \mathbf{W} \mathbf{v}_t \sim O(N^{-1})$, because elements of corresponding eigenvector $\mathbf{v}_t$ grow as $O(N^{-1/2})$ and only $K$ of those elements are multiplied by nonzero values to form each element of $\mathbf{W} \mathbf{v}_t$. In sum, the perturbation $\|\mathbf{E}\|_F$ grows while the eigenvalue $\lambda_t(\mathbf{W})$ shrinks. In linear embedding algorithms, the eigengap of interest is $\lambda_{gap} \doteq \lambda_1 - \lambda_0$. The tail eigenvalue $\lambda_0 = 0$ by construction but it is possible that $\lambda_0 > 0$ with numerical error, thus $\lambda_{gap} \leq \lambda_1$. Combining these facts, the ratio between the perturbation and the eigengap grows as $\|\mathbf{E}\|_F / \lambda_{gap} \sim O(N^{3/2})$ or faster. Now consider the shifted eigenproblem $\mathbf{I} - \mathbf{W}$ with leading (maximal) eigenvalues $1 - \lambda_0 \geq 1 - \lambda_1 \geq \cdots$ and unchanged eigenvectors. From matrix perturbation theory [10, thm. V.2.8], when $\mathbf{W}$ is perturbed to $\mathbf{W}' \doteq \mathbf{W} + \mathbf{E}$, the change in the leading eigenvalue from $1 - \lambda_0$ to $1 - \lambda_0'$ is bounded as $|\lambda_0' - \lambda_0| \leq \sqrt{2} \|\mathbf{E}\|_F$ and similarly $1 - \lambda_1' \leq 1 - \lambda_1 + \sqrt{2} \|\mathbf{E}\|_F$. Thus $\lambda_{gap}' \geq \lambda_{gap} - \sqrt{2} \|\mathbf{E}\|_F$. Since $\|\mathbf{E}\|_F / \lambda_{gap} \sim O(N^{3/2})$, the right hand side of the gap bound goes negative at a supralinear rate, implying that the eigenvalue ordering eventually becomes unstable with the possibility of the first and second eigenvalue/vector pairs being swapped. Mutual contamination of the eigenvectors happens well before: Under general (dense) conditions, the change in the eigenvector $\mathbf{v}_0$ is bounded as $\|\mathbf{v}_0' - \mathbf{v}_0\| \leq \frac{4 \|\mathbf{E}\|_F}{|\lambda_0 - \lambda_1| - \sqrt{2} \|\mathbf{E}\|_F}$ [10, thm. V.2.8]. (This bound is often tight enough to serve as a good approximation.) Specializing this to the sparse embedding matrix, we find that the bound weakens to $\|\mathbf{v}_0' - \mathbf{1} \cdot N^{-1/2}\| \sim \frac{O(\sqrt{N})}{O(N^{-1}) - O(\sqrt{N})} > \frac{O(\sqrt{N})}{O(N^{-1})} = O(N^{3/2})$.